# Reconstructing Stimulus Velocity from Neuronal Responses in Area MT

**Wyeth Bair, James R. Cavanaugh, J. Anthony Movshon**
Howard Hughes Medical Institute, and
Center for Neural Science
New York University
4 Washington Place, Room 809
New York, NY 10003
*wyeth@cns.nyu.edu, jamesc@cns.nyu.edu, tony@cns.nyu.edu*

## Abstract

We employed a white-noise velocity signal to study the dynamics of the response of single neurons in the cortical area MT to visual motion. Responses were quantified using reverse correlation, optimal linear reconstruction filters, and reconstruction signal-to-noise ratio (SNR). The SNR and lower bound estimates of information rate were lower than we expected. Ninety percent of the information was transmitted below 18 Hz, and the highest lower bound on bit rate was 12 bits/s. A simulated opponent motion energy subunit with Poisson spike statistics was able to out-perform the MT neurons. The temporal integration window, measured from the reverse correlation half-width, ranged from 30–90 ms. The window was narrower when a stimulus moved faster, but did not change when temporal frequency was held constant.

## 1  INTRODUCTION

Area MT neurons can show precise and rapid modulation in response to dynamic noise stimuli (Bair and Koch, 1996); however, computational models of these neurons and their inputs (Adelson and Bergen, 1985; Heeger, 1987; Grzywacz and Yuille, 1990; Emerson et al., 1992; Qian et al., 1994; Nowlan and Sejnowski, 1995) have primarily been compared to electrophysiological results based on time and ensemble averaged responses to deterministic stimuli, e.g., drifting sinusoidal gratings.

Using methods introduced by Bialek et al. (1991) and further analyzed by Gabbiani and Koch (1996) for the estimation of information transmission by a neuron about a white-noise stimulus, we set out to compare the responses of MT neurons for white-noise velocity signals to those of a model based on opponent motion energy sub-units.

The results of two analyses are summarized here. In the first, we compute a lower bound on information transmission using optimal linear reconstruction filters and examine the SNR as a function of temporal frequency. The second analysis examines changes in the reverse correlation (the cross-correlation between the stimulus and the resulting spike trains) as a function of spatial frequency and temporal frequency of the moving stimulus pattern.

## 2   EXPERIMENTAL METHODS

Spike trains were recorded extracellularly from 26 well-isolated single neurons in area MT of four anesthetized, paralyzed macaque monkeys using methods described in detail elsewhere (Levitt et al., 1994). The size of the receptive fields and the spatial and temporal frequency preferences of the neurons were assessed quantitatively using drifting sinusoidal gratings, after which a white-noise velocity signal, $s(t)$, was used to modulate the position (within a fixed square aperture) of a low-pass filtered 1D Gaussian white-noise (GWN) pattern. The frame rate of the display was 54 Hz or 81 Hz. The spatial noise pattern consisted of 256 discrete intensity values, one per spatial unit. Every 19 ms (or 12 ms at 81 Hz), the pattern shifted, or *jumped*, $\Delta$ spatial units along the axis of the neuron's preferred direction, where $\Delta$, the jump size, was chosen according to a Gaussian, binary, or uniform probability distribution. The maximum spatial frequency in the pattern was limited to prevent aliasing.

In the first type of experiment, 10 trials of a 30 s noise sequence, $s(t)$, and 10 trials of its reverse, $-s(t)$, were interleaved. A standard GWN spatial pattern and velocity modulation pattern were used for all cells, but for each cell, the stimulus was scaled for the receptive field size and aligned to the axis of preferred motion. Nine cells were tested with Gaussian noise at 81 Hz, 15 cells with binary noise at 81 Hz and 54 Hz, and 10 cells with uniform noise at 54 Hz.

In another experiment, a sinusoidal spatial pattern (rather than GWN) moved according to a binary white-noise velocity signal. Trials were interleaved with all combinations of four spatial frequencies at octave intervals and four *relative* jump sizes: 1/4, 1/8, 1/16, and 1/32 of each spatial period. Typically 10 trials of length 3 s were run. Four cells were tested at 54 Hz and seven at 81 Hz.

## 3   ANALYSIS AND MODELING METHODS

We used the linear reconstruction methods introduced by Bialek et al. (1991) and further analyzed by Gabbiani and Koch (1996) to compute an optimal linear estimate of the stimulus, $s(t)$, described above, based on the neuronal response, $x(t)$. A *single* neuronal response was defined as the spike train produced by $s(t)$ minus the spike train produced by $-s(t)$. This overcomes the neuron's limited dynamic range in response to anti-preferred direction motion (Bialek et al., 1991).

The linear filter, $h(t)$, which when convolved with the response yields the minimum mean square error estimate, $s_{est}$, of the stimulus can be described in terms of its Fourier transform,

$$H(\omega) = \frac{R_{sx}(-\omega)}{R_{xx}(\omega)}, \tag{1}$$

where $R_{sx}(\omega)$ is the Fourier transform of the cross-correlation $r_{sx}(\tau)$ of the stimulus and the resulting spike train and $R_{xx}(\omega)$ is the power spectrum, i.e., the Fourier transform of the auto-correlation, of the spike train (for details and references, see Bialek et al., 1991; Gabbiani and Koch, 1996). The noise, $n(t)$, is defined as the difference between the stimulus and the reconstruction,

$$n(t) = s_{est}(t) - s(t), \tag{2}$$

and the SNR is defined as

$$\text{SNR}(\omega) = \frac{R_{ss}(\omega)}{R_{nn}(\omega)}, \tag{3}$$

where $R_{ss}(\omega)$ is the Fourier power spectrum of the stimulus and $R_{nn}(\omega)$ is the power spectrum of the noise. If the stimulus amplitude distribution is Gaussian, then $\text{SNR}(\omega)$ can be integrated to give a lower bound on the rate of information transmission in bits/s (Gabbiani and Koch, 1996).

The motion energy model consisted of opponent energy sub-units (Adelson and Bergen, 1985) implemented with Gabor functions (Heeger, 1987; Grzywacz and Yuille, 1990) in two spatial dimensions and time. The spatial frequency of the Gabor function was set to match the spatial frequency of the stimulus, and the temporal frequency was set to match that induced by a sequence of jumps equal to the standard deviation (SD) of the amplitude distribution (which is the jump size in the case of a binary distribution). We approximated causality by shifting the output forward in time before computing the optimal linear filter. The model operated on the same stimulus patterns and noise sequences that were used to generate stimuli for the neurons. The time-varying response of the model was broken into two half-wave rectified signals which were interpreted as the firing probabilities of two units, a neuron and an anti-neuron that preferred the opposite direction of motion. From each unit, ten 30 s long spike trains were generated with inhomogeneous Poisson statistics. These 20 model spike trains were used to reconstruct the velocity signal in the same manner as the MT neuron output.

## 4   RESULTS

**Stimulus reconstruction.** Optimal linear reconstruction filters, $h(t)$, were computed for 26 MT neurons from responses to 30 s sequences of white-noise motion. A typical $h(t)$, shown in Fig. 1A (large dots), was dominated by a single positive lobe, often preceded by a smaller negative lobe. It was thinner than the reverse correlation $r_{sx}(\tau)$ (Fig. 1A, small dots) from which it was derived due to the division by the low-pass power spectrum of the spikes (see Eqn. 1). Also, $r_{sx}(\tau)$ occasionally had a slower, trailing negative lobe but did not have the preceding negative lobe of $h(t)$. On average, $h(t)$ peaked at -69 ms (SD 17) and was 33 ms (SD 12) wide at half-height. The peak for $r_{sx}(\tau)$ occurred at the same time, but the width was 41 ms (SD 15), ranging from 30–90 ms. The point of half-rise on the right side of the peak was -53 ms (SD 9) for $h(t)$ and -51 ms (SD 9) for $r_{sx}(\tau)$. For all plots,

vertical axes for velocity show normalized stimulus velocity, i.e., stimulus velocity was scaled to have unity SD before all computations.

Fig 1C (dots) shows the SNR for the reconstruction using the $h(t)$ in panel A. For 8 cells tested with *Gaussian* velocity noise, the integral of the log of the SNR gives a lower bound for information transmission, which was 6.7 bits/s (SD 2.8), with a high value of 12.3 bits/s. Most of the information was carried below 10 Hz, and 90% of the information was carried below 18.4 Hz (SD 2.1). In Fig. 1D, the failure of the reconstruction (dots) to capture higher frequencies in the stimulus (thick line) is directly visible. Both $h(t)$ and $\text{SNR}(\omega)$ were similar but on average slightly greater in amplitude for tests using binary and uniform distributed noise. Gaussian noise has many jumps at or near zero which may induce little or no response.

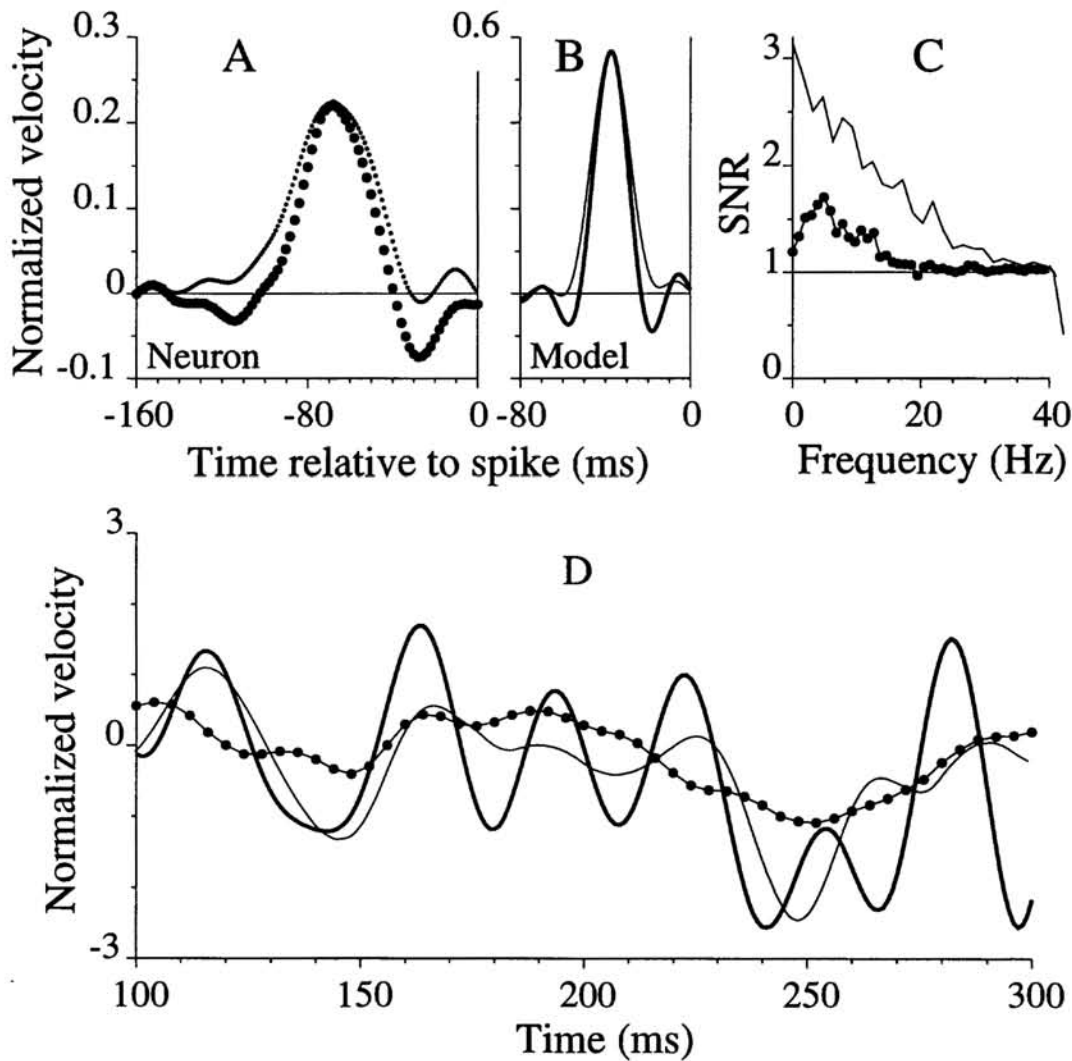

Figure 1: **(A)** Optimal linear filter $h(t)$ (big dots) from Eqn. 1 and cross-correlation $r_{sx}(\tau)$ (small dots) for one MT neuron. **(B)** $h(t)$ (thick line) and $r_{sx}(\tau)$ (thin line) for an opponent motion energy model. **(C)** $\text{SNR}(\omega)$ for the neuron (dots) and the model (line). **(D)** Reconstruction for the neuron (dots) and model (thin line) of the stimulus velocity (thick line). Velocity was normalized to unity SD. Curves for $r_{sx}(\tau)$ were scaled by 0.5. Note the different vertical scale in B.

An opponent motion energy model using Gabor functions was simulated with spatial SD 0.5°, spatial frequency 0.625 cyc/°, temporal SD 12.5 ms, and temporal frequency 20 Hz. The model was tested with a Gaussian velocity stimulus with SD 32°/s. Because an arbitrary scaling of the spatial parameters in the model does not affect the temporal properties of the information transmission, this was effectively the same stimulus that yielded the neuronal data shown in Fig. 1A. Spike trains were generated from the model at 20 Hz (matched to the neuron) and used to compute $h(t)$ (Fig. 1B, thick line). The model $h(t)$ was narrower than that for the MT neuron, but was similar to $h(t)$ for *V1 neurons* that have been tested (unpublished analysis). This simple model of a putative input sub-unit to MT transmitted 29 bits/s—more than the best MT neurons studied here. The SNR ratio and the reconstruction for the model are shown in Fig. 1C,D (thin lines). The filter $h(t)$ for the model (Fig. 1B thick line) was more symmetric than that for the neuron due to the symmetry of the Gabor function used in the model.

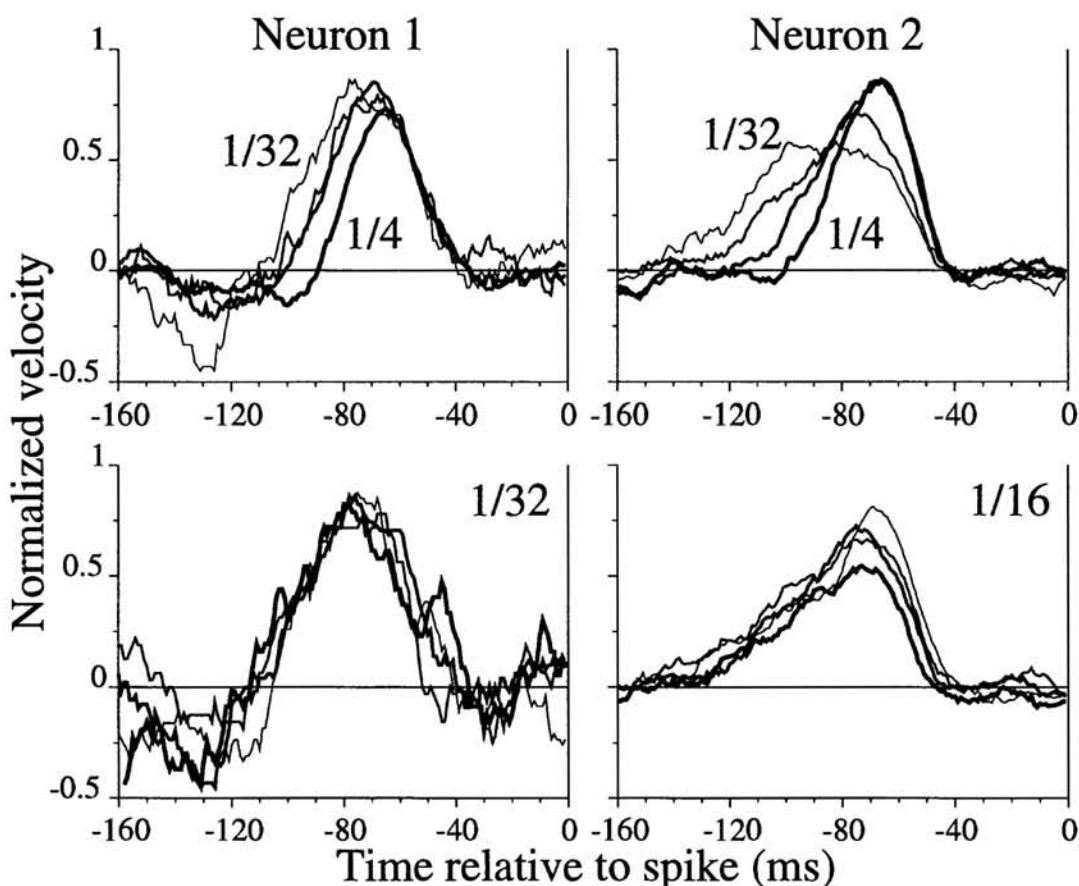

Figure 2: The width of $r_{sx}(\tau)$ changes with temporal frequency, but not spatial frequency. Data from two neurons are shown, one on the left, one on the right. **Top:** $r_{sx}(\tau)$ is shown for binary velocity stimuli with jump sizes 1/4, 1/8, 1/16, and 1/32 (thick to thin lines) of the spatial period (10 trials, 3 s/trial). The left side of the peak shifts leftward as jump size decreases. See text for statistics. **Bottom:** The relative jump size, thus temporal frequency, was constant for the four cases in each panel (1/32 on the left, 1/16 on the right). The peaks do not shift left or right as spatial frequency and jump size change inversely. Thicker lines represent larger jumps and lower spatial frequencies.

**Changes in** $r_{sx}(\tau)$. We tested 11 neurons with a set of binary white-noise motion stimuli that varied in spatial frequency and jump size. The spatial patterns were sinusoidal gratings. The peaks in $r_{sx}(\tau)$ and $h(t)$ were wider when smaller jumps (slower velocities) were used to move the same spatial pattern. Fig. 2 shows data for two neurons plotted for constant spatial frequency (top) and constant effective temporal frequency, or *contrast frequency* (bottom). Jump sizes were 1/4, 1/8, 1/16, and 1/32 (thick to thin lines, top panels) of the period of the spatial pattern. (Note, a 1/2 period jump would cause an ambiguous motion.) Relative jump size was constant in the bottom panels, but both the spatial period and the velocity increased in octaves from thin to thick lines. One of the plots in the upper panel also appears in the lower panel for each neuron. For 26 conditions in 11 MT neurons (up to 4 spatial frequencies per neuron) the left and right half-rise points of the peak of $r_{sx}(\tau)$ shifted leftward by 19 ms (SD 12) and 4.5 ms (SD 4.0), respectively, as jump size decreased. The width, therefore, increased by 14 ms (SD 12). These changes were statistically significant ($p < 0.001$, t-test). In fact, the left half-rise point moved leftward in all 26 cases, and in no case did the width at half-height decrease. On the other hand, there was no significant change in the peak width or half-rise times when temporal frequency was constant, as demonstrated in the lower panels of Fig. 2.

# 5  DISCUSSION

From other experiments using frame-based displays to present moving stimuli to MT cells, we know that roughly half of the cells can modulate to a 60 Hz signal in the preferred direction and that some provide reliable bursts of spikes on each frame but do not respond to null direction motion. Therefore, one might expect that these cells could easily be made to transmit nearly 60 bits/s by moving the stimulus randomly in either the preferred or null direction on each video frame. However, the stimuli that we employed here did not result in such high frequency modulation, nor did our best lower bound estimate of information transmission for an MT cell, 12 bits/s, approach the 64 bits/s capacity of the motion sensitive H1 neuron in the fly (Bialek et al., 1991). In recordings from seven V1 neurons (not shown here), two directional complex cells responded to the velocity noise with high temporal precision and fired a burst of spikes on almost every preferred motion frame and no spikes on null motion frames. At 53 frames/s, these cells transmitted over 40 bits/s. We hope that further investigation will reveal whether the lack of high frequency modulation in our MT experiments was due to statistical variation between animals, the structure of the stimulus, or possibly to anesthesia.

In spite of finding less high frequency bandwidth than expected, we were able to document consistent changes, namely narrowing, of the temporal integration window of MT neurons as temporal frequency increased. Similar changes in the time constant of motion processing have been reported in the fly visual system, where it appears that neither velocity nor temporal frequency alone can account for all changes (de Ruyter et al., 1986; Borst & Egelhaaf, 1987). The narrowing of $r_{sx}(\tau)$ with higher temporal frequency does not occur in our simple motion energy model, which lacks adaptive mechanisms, but it could occur in a model which integrated signals from many motion energy units having distributed temporal frequency tuning, even without other sources of adaptation.

We were not able to assess whether changes in the integration window developed quickly at the beginning of individual trials, but an analysis not described here at least indicates that there was very little change in the position and width of $r_{sx}(\tau)$ and $h(t)$ after the first few seconds during the 30 s trials.

## Acknowledgements

This work was funded by the Howard Hughes Medical Institute. We thank Fabrizio Gabbiani and Christof Koch for helpful discussion, Lawrence P. O'Keefe for assistance with electrophysiology, and David Tanzer for assistance with software.

## References

Adelson EH, Bergen JR (1985) Spatiotemporal energy models for the perception of motion. *J Opt Soc Am A* **2**:284–299.

Bair W, Koch C (1996) Temporal precision of spike trains in extrastriate cortex of the behaving macaque monkey. *Neural Comp* **8**:1185–1202.

Bialek W, Rieke F, de Ruyter van Steveninck RR, Warland D (1991) Reading a neural code. *Science* **252**:1854–1857.

Borst A, Egelhaaf M (1987) Temporal modulation of luminance adapts time constant of fly movement detectors. *Biol Cybern* **56**:209–215.

Emerson RC, Bergen JR, Adelson EH (1992) Directionally selective complex cells and the computation of motion energy in cat visual cortex. *Vision Res* **32**:203–218.

Gabbiani F, Koch C (1996) Coding of time-varying signals in spike trains of integrate-and-fire neurons with random threshold. *Neural Comp* **8**:44–66.

Grzywacz NM, Yuille AL (1990) A model for the estimate of local image velocity by cells in the visual cortex. *Proc R Soc Lond B* **239**:129–161.

Heeger DJ (1987) Model for the extraction of image flow. *J Opt Soc Am A* **4**:1455–1471.

Levitt JB, Kiper DC, Movshon JA (1994) Receptive fields and functional architecture of macaque V2. J.Neurophys. **71**:2517–2542.

Nowlan SJ, Sejnowski TJ (1994) Filter selection model for motion segmentation and velocity integration. *J Opt Soc Am A* **11**:3177–3200.

Qian N, Andersen RA, Adelson EH (1994) Transparent motion perception as detection of unbalanced motion signals .3. Modeling. *J Neurosc* **14**:7381–7392.

de Ruyter van Steveninck R, Zaagman WH, Mastebroek HAK (1986) Adaptation of transient responses of a movement-sensitive neuron in the visual-system of the blowfly calliphora-erythrocephala. *Biol Cybern* **54**:223–236.